# Oscillatory Model of Short Term Memory

**David Horn**
School of Physics and Astronomy
Raymond and Beverly Sackler
Faculty of Exact Sciences
Tel-Aviv University
Tel Aviv 69978, Israel

**Marius Usher***
Dept. of Applied Mathematics
and Computer Science
Weizmann Institute of Science
Rehovot 76100, Israel

## Abstract

We investigate a model in which excitatory neurons have dynamical thresholds which display both fatigue and potentiation. The fatigue property leads to oscillatory behavior. It is responsible for the ability of the model to perform segmentation, i.e., decompose a mixed input into staggered oscillations of the activities of the cell-assemblies (memories) affected by it. Potentiation is responsible for sustaining these staggered oscillations after the input is turned off, i.e. the system serves as a model for short term memory. It has a limited STM capacity, reminiscent of the magical number $7 \pm 2$.

## 1 Introduction

The limited capacity ($7 \pm 2$) of the short term memory (STM) has been a subject of major interest in the psychological and physiological literature. It seems quite natural to assume that the limited capacity is due to the special dynamical nature of STM. Recently, Crick and Koch (1990) suggested that the working memory is functionally related to the binding process, and is obtained via synchronized oscillations of neural populations. The capacity limitation of STM may then result from the competition between oscillations representing items in STM. In the model which we investigate this is indeed the case.

Models of oscillating neural networks can perform various tasks:

1. Phase-locking and synchronization in response to global coherence in the stimuli, such as similarity of orientation and continuity (Kamen et al. 1989; Sompolinsy et al. 1990; Konig & Schillen 1991).

2. Segmentation of incoherent stimuli in low level vision via desynchronization, using oscillator networks with delayed connections (Schillen & Konig 1991).

3. Segmentation according to semantic content, i.e., separate an input of mixed information into its components which are known memories of the system (Wang et al. 1990, Horn and Usher 1991). In these models the memories are represented by competing cell assemblies. The input, which affects a subset of these assemblies, induces staggered oscillations of their activities. This works as long as the number of memories in the input is small, of the order of 5.

4. Binding, i.e., connecting correctly different attributes of the same object which appear in the mixed input (Horn et al. 1991). Binding can be interpreted as matching the phases of oscillations representing attributes of the same object in two different networks which are coupled in a way which does not assume any relation between the attributes.

To these we add here the important task of

5. STM, i.e., keeping information about segmentation or binding after the input is turned off.

In order to qualify as models for STM, the staggered oscillations have to prevail after the input stimuli disappear. Unfortunately, this does not hold for the models quoted above. Once the input disappears, either the network's activity dies out, or oscillations of assemblies not included in the original input are turned on. In other words, the oscillations have no inertia, and thus they do not persist after the disappearance of the sensory input. Our purpose is to present a model of competing neural assemblies which, upon receiving a mixed input develops oscillations which prevail after the stimulus disappears. In order to achieve this, the biological mechanism of post tetanic potentiation will be used.

## 2    Dynamics of Short Term Potentiation

It was shown that following a tetanus of electrophysiological stimulation temporary modifications in the synaptic strengths, mostly non Hebbian, are observed (Crick and Koch, 1990; Zucker, 1989). The time scale of these synaptic modifications ranges between 50 ms to several minutes. A detailed description of the processes responsible for this mechanism was given by Zucker (1989), exhibiting a rather complex behavior. In the following we will use a simplified version of these mechanisms involving two processes with different time scales. We assume that following a prolonged activation of a synapse, the synaptic strength exhibits depression on a short time scale, but recovers and becomes slightly enhanced on a longer time scale. As illustrated in Fig 1 of Zucker (1989), this captures most of the dynamics of Short Term Potentiation. The fact that these mechanisms are non Hebbian implies that all synapses associated with a presynaptic cell are affected, and thus the unit of change is the presynaptic cell (Crick & Koch 1990).

Our previous oscillatory neural networks were based on the assumption that, in addition to the customary properties of the formal neuron, its threshold increases when the neuron keeps firing, thus exhibiting adaptation or fatigue (Horn & Usher 1989). Motivated by the STP findings we add a new component of facilitaion, which takes place on a longer time scale than fatigue. We denote the dynamical threshold by the continuous variable $r$ which is chosen as a sum of two components, $f$ and $p$, representing fatigue and potentiation,

$$r = a_1 f - a_2 p. \tag{1}$$

Their dynamics is governed by the equations

$$\gamma df/dt = m + (1/c_1 - 1)f \qquad \gamma dp/dt = m + (1/c_2 - 1)p \tag{2}$$

where $m$ describes the average neuron activity (firing rate) on a time scale which is large compared to the refractory period. The time constants of the fatigue and potentiation components, $\tau_i = \frac{c_i}{c_i - 1}$ are chosen so that $\tau_1 < \tau_2$. As a result the neuron displays fatigue on a short time scale, but recovers and becomes slightly enhanced (potentiated) on a longer time scale. This is clearly seen in Fig. 1, which shows the behavior when the activity $m$ of the corresponding neuron is clamped at 1 for some time (due to sensory input) and quenched to zero afterwards.

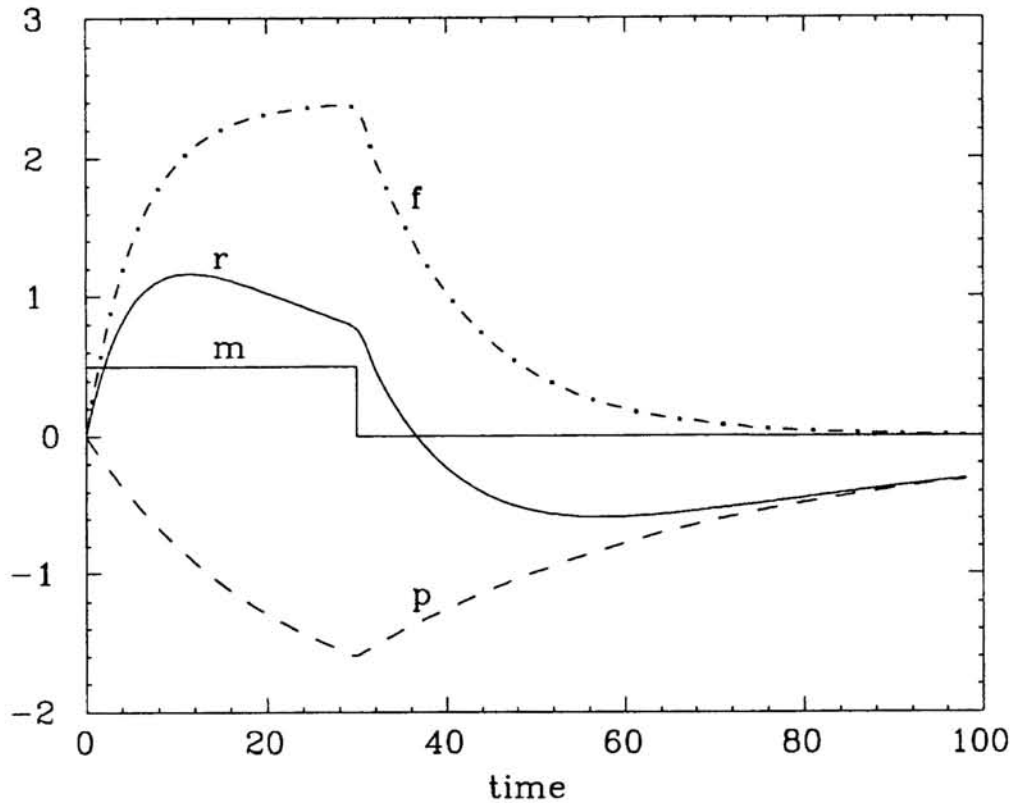

Figure 1: Behavior of the dynamic threshold $r$ and its fatigue $f$ and potentiation $p$ components, when the neuron activity $m$ is clamped as shown. Time scale is arbitrary. The parameters are $c_1 = 1.2$ $c_2 = 1.05$ $a_1 = 4$ $a_2 = 1$ .

We observe here that the threshold increases during the cell's activation, being driven to its asymptotic value $a_1 \frac{c_1 - 1}{c_1}$. After the release of the stimulus the dynamic threshold decreases (i.e. the neuron recovers) and turns negative (signifying

potentiation). The parameters were chosen so that asymptotically the threshold reaches zero, i.e. no permanent effect is left. In our model we will assume a similar behavior for the excitatory cell-assemblies which carry the memories in our system.

## 3    The Model

Our basic model (Horn & Usher 1990) is composed of two kinds of neurons which are assumed to have excitatory and inhibitory synapses exclusively. Memory patterns are carried by excitatory neurons only. Furthermore, we make the simplifying assumption that the patterns do not overlap with one another, i.e. the model is composed of disjoint Hebbian cell-assemblies of excitatory neurons which affect one another through their interaction with a single assembly of inhibitory neurons.

Let us denote by $m^\mu(t)$ the fraction of cell-assembly number $\mu$ which fires at time $t$, and by $m^I(t)$ the fraction of active inhibitory neurons. We will refer to $m^\mu$ as the activity of the $\mu$th memory pattern. There are $P$ different memories in the model, and their activities obey the following differential equations

$$dm^\mu/dt = -m^\mu + F_T(Am^\mu - Bm^I - \theta^\mu + i^\mu) \tag{3}$$

$$dm^I/dt = -m^I + F_T(CM - Dm^I - \theta^I)$$

where

$$M = \sum_\mu m^\mu \qquad F_T(x) = (1 + e^{-x/T})^{-1}. \tag{4}$$

$\theta^\mu$ and $\theta^I$ are the thresholds of all excitatory and inhibitory neurons correspondingly and $i^\mu$ represents the input into cell assembly $\mu$. The four parameters $A\,B\,C$ and $D$ are all positive and represent the different couplings between the neurons. This system is an attractor neural network. In the absence of input and dynamical thresholds it is a dissipative system which flows into fixed points determined by the memories.

This system is a generalization of the E-I model of Wilson and Cowan (1972) in which we have introduced competing memory patterns. The latter make it into an attractor neural network. Wilson and Cowan have shown that a pair of excitatory and inhibitory assemblies, when properly connected, will form an oscillator. We induce oscillations in a different way, keeping the option of having the network behave either as an attractor neural network or as an oscillating one: we turn the thresholds of the excitatory neurons into dynamic variables, which are defined by

$$\theta^\mu = \theta_0^\mu + br^\mu.$$

The dynamics of the new variables $r^\mu$ are chosen to follow equations (1) and (2) where all elements, $r\ f\ p$ and $m$ refer to the same cell-assembly $\mu$. To understand the effects of this change let us first limit ourselves to the fatigue component only, i.e. $a_1 = 1$ and $a_2 = 0$ in Eq. 1. Imagine a situation in which the system would flow into a fixed point $m^\mu = 1$. $r^\mu$ will then increase until it reaches the value $c_1/(c_1-1)$. This means that the argument of the $F_T$ function in the equation for $m^\mu$ decreases by $g = bc_1/(c_1 - 1)$ . If this overcomes the effect of the other terms the amplitude $m^\mu$ decreases and the system moves out of the attractor and falls into the basin of a different center of attraction. This process can continue indefinitely creating

an oscillatory network which moves from one memory to another. Envisage now turning on a $p^\mu$ component leading to an $r^\mu$ behavior of the type depicted in Fig. 1. Its effect will evidently be the same as the input $i^\mu$ in Eq. (3) during the time in which it is active. In other words, it will help to reactivate the cell-assembly $\mu$, thus carrying the information that this memory was active before. Therefore, its role in our system is to serve as the inertia component necessary for creating the effect of STM.

## 4    Segmentation and Short Term Memory

In this section we will present results of numerical investigations of our model. The parameters used in the following are $A = C = D = 1\ B = 1.1\ \theta_0^\mu = 0.075\ \theta^I = 0.55\ T = 0.05\ b = 0.2\ \gamma = 2.5$ and the values of $a_i$ and $c_i$ of Fig. 1. We let $n$ of the $P$ memories have a constant input of the form

$$i^\mu = i \quad \mu = 1, \cdots, n \qquad i^\mu = 0 \quad \mu = n+1, \cdots, P. \tag{5}$$

An example of the result of a system with $P = 10$ and $n = 4$ is shown in Fig. 2.

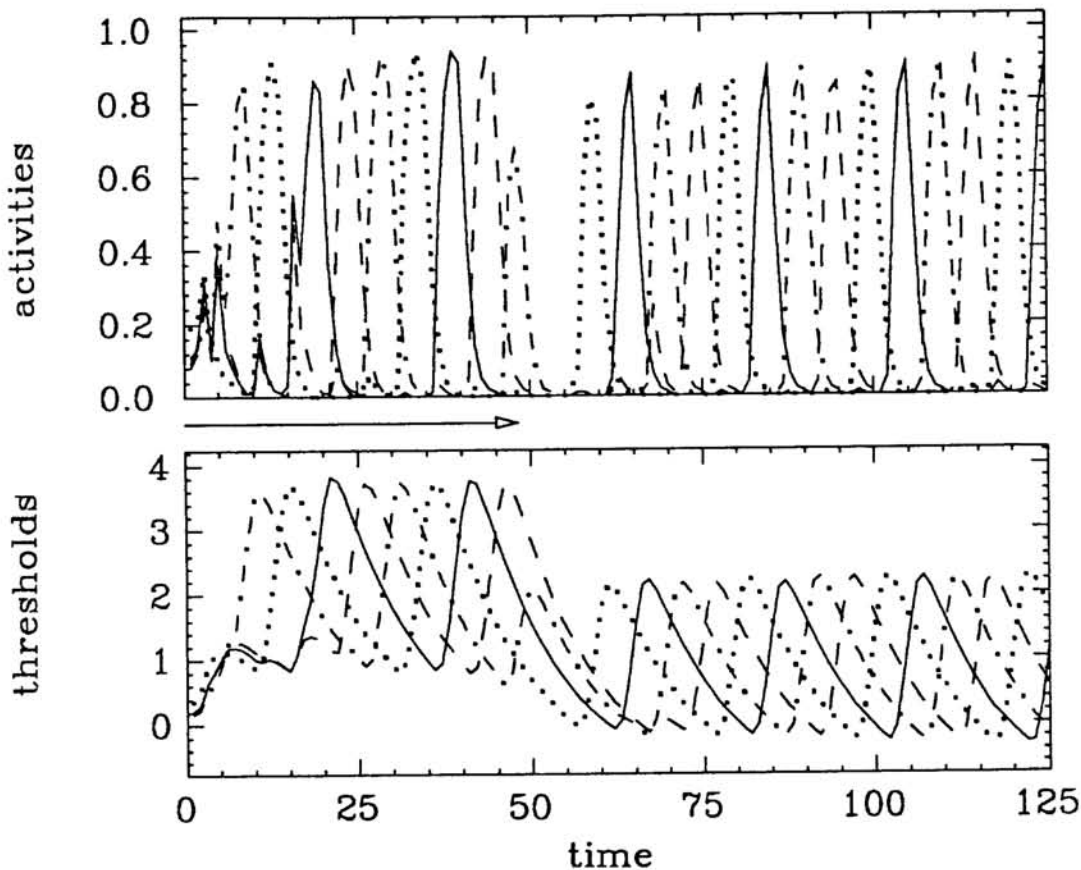

Figure 2: Results of our model for $P = 10$ memories and $n = 4$ inputs. The first frame displays the activities $m$ of the four relevant cell-assemblies, and the second frame represents their $r$ values. The arrow indicates the duration of the mixed input.

Here we display the activities of the cell-assemblies that receive the constant input and their corresponding average thresholds. While the signal of the mixed input is on (denoted by an arrow along the time scale) we see how the phenomenon of segmentation develops. The same staggered oscillation of the four cell-assemblies which received an input is sustained after the signal is turned off. This indicates that the system functions as a STM. Note that no synaptic connections were changed and, once the system will receive a new input its behavior will be revised. However, as long as it is left alone, it will continue to activate the cell-assemblies affected by the original input.

We were able to obtain good results only for low $n$ values, $n \leq 4$. As $n$ is increased we have difficulties with both segmentation and STM. By modifying slightly the paradigm we were able to feed 5 different inputs in a STM, as shown in Fig. 3. This required presenting them at different times, as indicated by the 5 arrows on this figure. In other words, this system does not perform segmentation but it continues to work as a STM. Note, however, that the order of the different activities is no longer maintained after the stimuli are turned off.

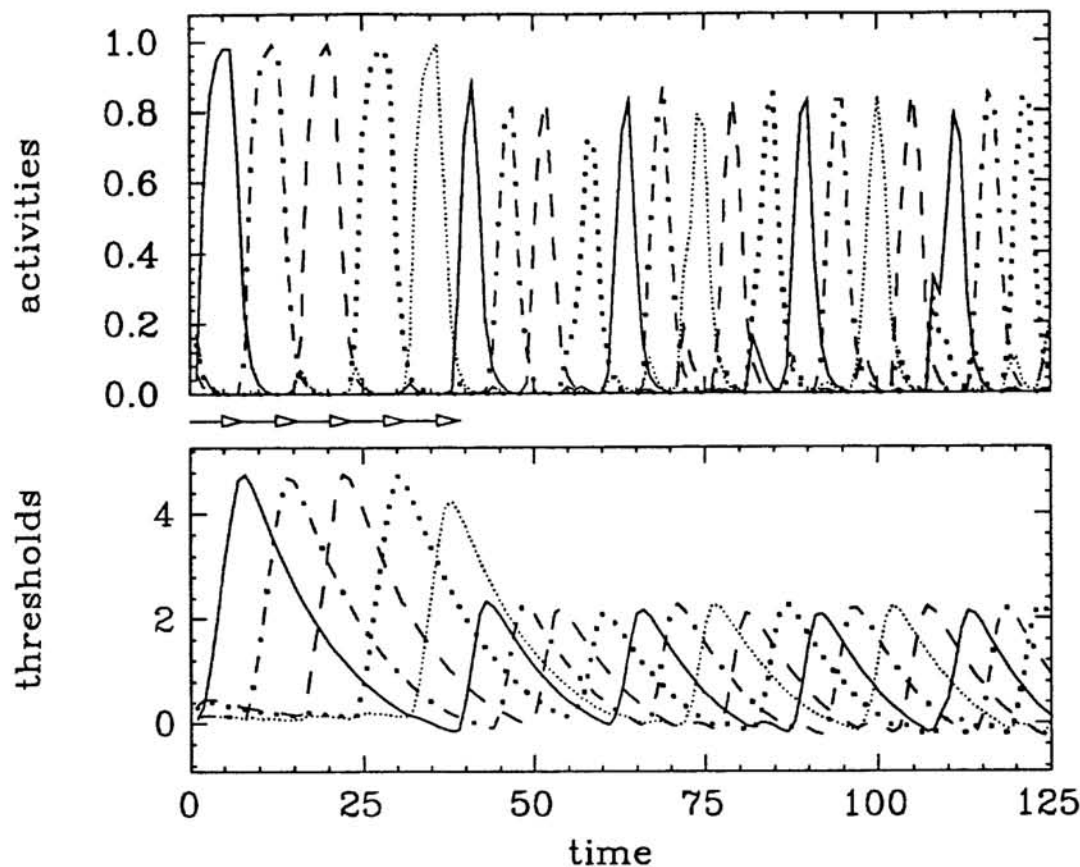

Figure 3: Results for 5 inputs which are fed in consecutively at the times indicated by the short arrows. The model functions as STM without segmentation.

# 5    Discussion.

Psychological experiments show that subjects can repeat a sequence of verbal items in perfect order as long as their number is small ($7 \pm 2$). The items may be numbers or letters but can also be combinations of the latter such as words or recognizable dates or acronyms. This proves that STM makes use of the encoded material in the long term memory (Miller 1956). This relation between the two different kinds of memory lies in the basis of our model. Long term memory is represented by excitatory cell assemblies. Incorporating threshold fatigue into the model, it acquires the capability of performing temporal segmentation of external input. Adding to the threshold post tetanic potentiation, the model becomes capable of maintaining the segmented information in the form of staggered oscillations. This is the property which we view as responsible for STM.

Both segmentation and STM have very limited capacities. This seems to follow from the oscillatory nature of the system which we use to model these functions. In contrast with long term memory, whose capacity can be increased endlessly by adding neurons and synaptic connections, we find here that only a few items can be stored in the dynamic fashion of staggered oscillations, irrespective of the size of the system. We regard this result as very significant, in view of the fact that the same holds for the limited psychological ability of attention and STM. It may indicate that the oscillatory model contains the key to the understanding of these psychological findings.

In order to validate the hypothesis that STM is based on oscillatory correlations between firing rates of neurons, some more experimental neurobiological and psychophysical research is required. While no conclusive results were yet obtained from recordings of the cortical activity in the monkey, some positive support has been obtained in psychophysical experiments. Preliminary results show that an oscillatory component can be found in the percentage of correct responses in STM matching experiments (Usher & Sagi 1991).

Our mathematical model is based on many specific assumptions. We believe that our main results are characteristic of a class of such models which can be obtained by changing various elements in our system. The main point is that dynamical storage of information can be achieved through staggered oscillations of memory activities. Moreover, to sustain them in the absence of an external input, a potentiation capability has to be present. A model which contains both should be able to accomodate STM in the fashion which we have demonstrated.

### Acknowledgements

M. Usher is the recipient of a Dov Biegun post-doctoral fellowship. We wish to thank S. Popescu for helpful discussions.

## Footnotes

*Present address: Division of Biology, 216-76, Caltech, Pasadena CA 91125.

### References

Crick,F. & Koch,C. 1990. Towards a neurobiological theory of consciousness. *Seminars in the Neurosciences 2*, 263–275.

Horn,D., Sagi,D. & Usher,M. 1991. Segmentation, binding and illusory conjunctions. *Neural Comp. 3*, 509–524.

Horn,D. & Usher,M. 1989. Neural networks with dynamical thresholds, *Phys. Rev. A 40*, 1036–1044.

Horn,D. & Usher,M. 1990. Excitatory-inhibitory networks with dynamical thresholds, *Int. J. Neural Syst. 1*, 249–257.

Horn,D. & Usher,M. 1991. Parallel Activation of Memories is an Oscillatory Neural Network. *Neural Comp. 3*, 31–43.

Kammen,D.M., Holmes,P.J. & Koch C. 1990. Origin of oscillations in visual cortex: Feedback versus local coupling. In *Models of Brain Function*, M.Cotterill ed., pp 273–284. Cambridge University Press.

Konig,P. & Schillen,T.B. 1991. Stimulus-dependent assembly formation of oscillatory responses: I. Synchronization, *Neural Comp. 3*, 155–166.

Miller,G. 1956. The magical number seven plus minus two. *Psych. Rev., 63*, 81–97.

Sompolinsky,H., Golomb,D. & Kleinfeld,D. 1990. Global processing of visual stimuli in a neural network of coupled oscillators. *Proc. Natl. Acad. of Sci. USA, 87*, 7200–7204.

Schillen,T.B. & Konig,P. 1991. Stimulus-dependent assembly formation of oscillatory responses: I. Synchronization, *Neural Comp. 3*, 155–166.

Wang,D., Buhmann,J. & von der Malsburg,C. 1990. Pattern segmentation in associative memory. *Neural Comp. 2*, 94–106.

Wilson,H.R. & Cowan,J.D. 1972. Excitatory and inhibitory interactions in localized populations of model neurons. *Biophys. J. 12*, 1–24.

Usher,M. & Sagi D. 1991, in preparation.

Zucker,R.S. 1989. Short-term synaptic plasticity. *Ann. Rev. Neurosci. 12*, 13–31.
